# Pruning with generalization based weight saliencies: $\gamma$OBD, $\gamma$OBS

**Morten With Pedersen**
**Lars Kai Hansen**
**Jan Larsen**
CONNECT, Electronics Institute
Technical University of Denmark B349
DK-2800 Lyngby, DENMARK
emails: with,lkhansen,jlarsen@ei.dtu.dk

## Abstract

The purpose of most architecture optimization schemes is to improve generalization. In this presentation we suggest to estimate the weight saliency as the associated change in generalization error if the weight is pruned. We detail the implementation of both an $O(N)$-storage scheme extending OBD, as well as an $O(N^2)$ scheme extending OBS. We illustrate the viability of the approach on prediction of a chaotic time series.

## 1 BACKGROUND

Optimization of feed-forward neural networks by pruning is a well-established tool, used in many practical applications. By careful fine tuning of the network architecture we may improve generalization, decrease the amount of computation, and facilitate interpretation.

The two most widely used schemes for pruning of feed-forward nets are: Optimal Brain Damage (OBD) due to (LeCun et al., 90) and the Optimal Brain Surgeon (OBS) (Hassibi et al., 93). Both schemes are based on weight ranking according to *saliency* defined as the change in *training error* when the particular weight is pruned. In OBD the saliency is estimated as the direct change in training error, i.e., without retraining of the remaining weights, while the OBS scheme includes retraining in a local quadratic approximation. The *rationale* of both methods is that if the least significant weights (according to training error) are deleted, we gracefully relieve the danger of overfitting. However, in both cases one clearly needs a *stop criterion*. As both schemes aim at minimal generalization error an estimator for this quantity is needed. The most obvious candidate estimate is a *test error* estimated on a validation set. Validation sets, unfortunately, are notoriously very noisy (see,

e.g., the discussion in Weigend et al., 1990). Hence, an attractive alternative is to estimate the test error by statistical means, e.g., Akaike's FPE (Akaike, 69). For regression type problems such a pruning stop criterion was suggested in (Svarer et al., 93).

*However, why not let the saliency itself reflect the possible improvement in test error? This is the idea that we explore in this contribution.*

## 2 GENERALIZATION IN REGULARIZED NEURAL NETWORKS

The basic asymptotic estimate of the generalization error was derived by Akaike (Akaike, 1969); the so-called Final Prediction Error (FPE). The use of FPE-theory for neural net learning has been pioneered by Moody (see e.g. (Moody, 91)), who derived estimators for the average generalization error in regularized networks.

Our network is a feed-forward architecture with $n_I$ input units, $n_H$ hidden sigmoid units and a single linear output unit, appropriate for scalar function approximation. The initial network is fully connected between layers and implements a non-linear mapping from input space $\mathbf{x}(k)$ to the real axis: $\widehat{y}(k) = F_{\mathbf{u}}(\mathbf{x}(k))$, where $\mathbf{u} = [\mathbf{w}, \mathbf{W}]$ is the $N$-dimensional weight vector and $\widehat{y}(k)$ is the prediction of the target output $y(k)$. The particular family of non-linear mappings considered can be written as:

$$F_{\mathbf{u}}(\mathbf{x}(k)) = \sum_{j=1}^{n_H} W_j \tanh\left(\sum_{i=1}^{n_I} w_{ji} x_i(k) + w_{j0}\right) + W_0, \tag{1}$$

$W_j$ are the hidden-to-output weights while $w_{ij}$ connect the input and hidden units.

We use the sum of squared errors to measure the network performance

$$E_{\text{train}} = \frac{1}{p} \sum_{k=1}^{p} [y(k) - F_{\mathbf{u}}(\mathbf{x}(k))]^2, \tag{2}$$

where $p$ is the number of training examples. To ensure numerical stability and to assist the pruning procedure we augment the cost function with a regularization term.[1] The resulting cost function reads

$$E = E_{\text{train}} + \frac{1}{2}\mathbf{u}^T \mathbf{R}\mathbf{u} \tag{3}$$

The main source of uncertainty in learning is the shortage of training data. Fitting the network from a finite set of noisy examples means that the noise in these particular examples will be fitted as well and when presented with a new test example the network will make an error which is larger than the error of the "optimal network" trained on an infinite training set. By careful control of the fitting capabilities, e.g., by pruning, such overfitting may be reduced.

The generalization error is defined as the average squared error on an example from the example distribution function $P(\mathbf{x}, y)$. The examples are modeled by a *teacher* network with weights $\mathbf{u}^*$, degraded by additive noise: $y(k) = F_{\mathbf{u}^*}(\mathbf{x}(k)) + \nu(k)$. The noise samples $\nu(k)$ are independent identically distributed variables with finite, but unknown variance $\sigma^2$. Further, we assume that the noise terms are independent of the corresponding inputs. The quantity of interest for model optimization is the training set average of the generalization error, viz., the average over an ensemble

of networks in which each network is provided with its individual training set. This averaged generalization error is estimated by

$$\widehat{E}_{\text{test}} = \left(1 + \frac{N_{\text{eff}}}{p}\right) \sigma^2 + O\left((1/p)^2\right), \tag{4}$$

with the effective number of parameters being $N_{\text{eff}} = \text{tr}(\mathbf{H}\mathbf{J}^{-1}\mathbf{H}\mathbf{J}^{-1})$ (Larsen and Hansen, 94). The Hessian, $\mathbf{H}$, is the second derivative matrix of the training error with respect to the weights and thresholds, while $\mathbf{J}$ is the regularized Hessian: $\mathbf{J} = \mathbf{H} + \mathbf{R}$. An asymptotically unbiased estimator of the noise level is provided by: $\sigma^2 = E_{\text{train}}/(1 - N_{\text{eff}}/p)$. Inserting, we get

$$\widehat{E}_{\text{test}} = \frac{p + N_{\text{eff}}}{p - N_{\text{eff}}} E_{\text{train}} \approx \left(1 + \frac{2N_{\text{eff}}}{p}\right) E_{\text{train}}. \tag{5}$$

While OBD and OBS are based on estimates of the change in $E_{\text{train}}$ we see that in order to obtain saliencies that estimate the change in generalization we must generally take the prefactor into account. We note that if the network is *not* regularized $N_{\text{eff}} = \text{tr}(\mathbf{H}\mathbf{J}^{-1}\mathbf{H}\mathbf{J}^{-1}) = \text{tr}(\mathbf{1}) = N$, in which case the prefactor is only a function of the total number of weights. In this case ranking according to training error saliency is equivalent to ranking according to generalization error.

However, in the generic case of a regularized network this is no more true $(N_{\text{eff}} < N)$, and we need to evaluate the change in the prefactor, i.e., in the effective number of parameters, associated with pruning a weight. Denoting the generalization based saliency of weight $u_l$ as $E_{\text{test},l}$, we find

$$\delta E_{\text{test},l} \approx \delta E_{\text{train},l} - \frac{2\left(N_{\text{eff}} - N_{\text{eff},l}\right)}{p} E_{\text{train}} \tag{6}$$

Where the number of parameters after pruning of weight $l$ is $N_{\text{eff},l}$, and $\delta E_{\text{train},l}$ is the training error based saliency.

To proceed we outline two implementations, the major difference being the computational complexity involved. In the first, which is an elaboration on the OBD scheme, the storage complexity is proportional to the number of weights and thresholds $(N)$, while in the second scheme the complexity scales with $N^2$, and is a generalization of the OBS. To emphasize that we use the generalization error for ranking of weights we use the prefix γ: γOBD and γOBS.

## 3   γOBD: AN $O(N)$ IMPLEMENTATION

Our $O(N)$ simulator is based on *batch mode*, second order pseudo-Gauss Newton optimization which is described in (Svarer et al., 93). The scheme, being based on the diagonal approximation for the Hessian, requires storage of a number of variables scaling linearly with the number of parameters $N$. As in (Le Cun et al., 90) we approximate the second derivative matrix by the positive semi-definite expression:

$$\frac{\partial^2 E_{\text{train}}}{\partial u_j^2} \approx \frac{2}{p} \sum_{k=1}^{p} \left(\frac{\partial F_{\mathbf{u}}(\mathbf{x}(k))}{\partial u_j}\right)^2. \tag{7}$$

In the diagonal approximation we find

$$N_{\text{eff}} = \sum_{j=1}^{N} \left(\frac{\lambda_j}{\lambda_j + \alpha_j/p}\right)^2, \tag{8}$$

where $\lambda_j \equiv \partial^2 E_{\text{train}}/\partial u_j^2$. Further, $\alpha_j/p$ are the weight decay parameters (diagonal elements of the regularization matrix $\mathbf{R}$).

The OBD method proposed by (Le Cun et al., 90) was successfully applied to reduce large networks for recognition of handwritten digits. The basic idea is to estimate the increase in the *training error* when deleting weights. Expanding the training error to second order in the pruned weight magnitude it is found that

$$\delta E_{\text{train},l} = \left( \frac{\alpha_l}{p} + \frac{1}{2}\frac{\partial^2 E_{\text{train}}}{\partial u_l^2} \right) u_l^2. \tag{9}$$

This estimate takes into account that the weight decay terms force the weights to depart from the minimum of the training set error. The first derivative of the training error is non-zero, hence, the first term in (9). Computationally, we note that the diagonal Hessian terms are reused from the pseudo Gauss-Newton training scheme.

Using (6) and the diagonal form of $N_{\text{eff}}$, we find the following approximative expression for generalization saliency ($\gamma$OBD):

$$\delta E_{\text{test},l} \approx \delta E_{\text{train},l} - \frac{2}{p}\left( \frac{\lambda_l}{\lambda_l + \alpha_l/p} \right)^2 E_{\text{train}} \tag{10}$$

From this expression we learn that of two weights inducing similar changes in training error we should delete the one which has the largest ratio of training error curvature ($\lambda$) to weight decay, i.e., the weight which has been *least* influenced by weight decay. However, from a computational point of view we also want to reduce the number of parameters as far as possible; so we might in fact accept to delete weights with small positive generalization saliency (in particular considering the amount of approximation involved in the estimates).

## 4   $\gamma$OBS: AN O($N^2$) IMPLEMENTATION

In the Optimal Brain Surgeon (Hassibi et al., 92) the increase in training error is estimated including the effects of quadratic retraining. This allows for pruning of more general degrees of freedom, e.g., situations where the training error induces linear constraints among two or more weights. The price to be paid is that we need to operate with the full $N \times N$ Hessian matrix of second derivatives. The O($N^2$) simulator, hence, is based on full Gauss Newton optimization. When eliminating the $l$'th weight retraining is determined by

$$\delta\mathbf{u}_l = -\frac{u_l}{(\mathbf{J}^{-1})_{ll}}\mathbf{J}^{-1}\mathbf{e}_l \tag{11}$$

where $\mathbf{e}_l$ is the $l$'th unit vector. We need to modify the OBS saliencies when working from a weight decay regularized cost function. The modified saliencies were given in (Hansen and With, 94)[2]

$$\delta E_{train,l} = \frac{1}{2}\frac{u_l^2}{(\mathbf{J}^{-1})_{ll}} + \frac{\alpha}{p}\left( \frac{u_l(\mathbf{e}_l^T\mathbf{J}^{-1}\mathbf{u})}{(\mathbf{J}^{-1})_{ll}} - \frac{1}{2}\frac{u_l^2(\mathbf{J}^{-2})_{ll}}{((\mathbf{J}^{-1})_{ll})^2} \right) \tag{12}$$

Whether using the generalization based $\gamma$OBS or standard OBS, we want to point to an important aspect of OBS that seems not to be generally appreciated, namely the

problem of "nuisance" parameters (White, 89), (Larsen, 93). When eliminating an output weight $u_o$, all the weights to the corresponding hidden unit are in effect also pruned away. Such a situation is well-known in the statistics literature on model selection where such "ghost" input weights are known as nuisance parameters. It is important to remove these parameters from the network function before estimating the saliency $\delta E_{train,o}$ and the resulting effective number of parameters $N_{eff}$, as they would otherwise give "spurious" contributions to these estimates. Applying OBS without taking this fact into consideration often results in sudden jumps in the level of the network error due to pruning of an important weight based on a corrupted saliency estimate. Removing the superfluous weights from the weight vector $\mathbf{u}$ and the corresponding rows and columns in $\mathbf{J}$ to form the reduced (regularized) Hessian $\mathbf{J}_1$ is straightforward, but it is computationally expensive to invert each of the resulting (sub-)matrices $\mathbf{J}_1$ for use in (11) and (12). This cost can be considerably reduced by rearranging the rows and columns of $\mathbf{J}$ as

$$\mathbf{J} = \begin{bmatrix} \mathbf{J}_1 & \mathbf{J}_2 \\ \mathbf{J}_3 & \mathbf{J}_4 \end{bmatrix} \quad \rightarrow \quad \mathbf{J}^{-1} = \begin{bmatrix} (\mathbf{J}^{-1})_1 & (\mathbf{J}^{-1})_2 \\ (\mathbf{J}^{-1})_3 & (\mathbf{J}^{-1})_4 \end{bmatrix} \tag{13}$$

where $\mathbf{J}_2$, $\mathbf{J}_3$ and $\mathbf{J}_4$ are the rows and columns corresponding to the nuisance parameters. Using a standard lemma for partitioned matrices, we obtain

$$(\mathbf{J}_1)^{-1} = (\mathbf{J}^{-1})_1 - (\mathbf{J}^{-1})_2[(\mathbf{J}^{-1})_4]^{-1}(\mathbf{J}^{-1})_3 \tag{14}$$

which only calls for inversion of the (small) submatrix $(\mathbf{J}^{-1})_4$. In (Hassibi et al., 93) it was argued that one might save on computation by using an iterative scheme for calculation of the inverse Hessian $\mathbf{J}^{-1}$. However, since standard matrix inversion is an $O(N^3)$ operation while the iterative scheme scales as $O(pN^2)$, a detailed count shows that that it is only beneficial to use the iterative scheme in the atypical case $N > p/2$.

## 5  EXPERIMENT

We will illustrate the viability of the proposed methods on a standard problem of nonlinear dynamics viz. the Mackey-Glass chaotic time series. The series is generated by integration of the differential equation

$$\frac{dz(t)}{dt} = -bz(t) + a\frac{z(t-\tau)}{1 + z(t-\tau)^{10}} \tag{15}$$

where the constants are $a = 0.2$, $b = 0.1$ and $\tau = 17$. The series is resampled with sampling period 1 according to standard practice. The network configuration is $n_I = 6$, $n_H = 10$ and we train to implement a six step ahead prediction. That is, $\mathbf{x}(k) = [z(k-6), z(k-12), \cdots, z(k-6n_I)]$ and $y(k) = z(k)$. In Fig. 1 we show pruning scenarios based on the two different implementations. The training errors, test errors and FPE errors are plotted for a training set size of 250 examples, the test set comprises 8500 examples. In the left panel we show the results of pruning according to γOBD and similarly in the right panel we show the results of pruning as it occurred using γOBS. In this example we do not find significant improvement in performance by use of γOBS.

To illustrate the ability of the estimators for predicting the effects of pruning on the test error we plot in figure 2 the estimated test errors versus the actual test errors after pruning. In the OBD case this means the test error resulting from pruning the parameters without retraining, while in the OBS case it means the test error following pruning and retraining in the quadratic approximation. We note that the γOBD estimates of the test error approximately equal the actual

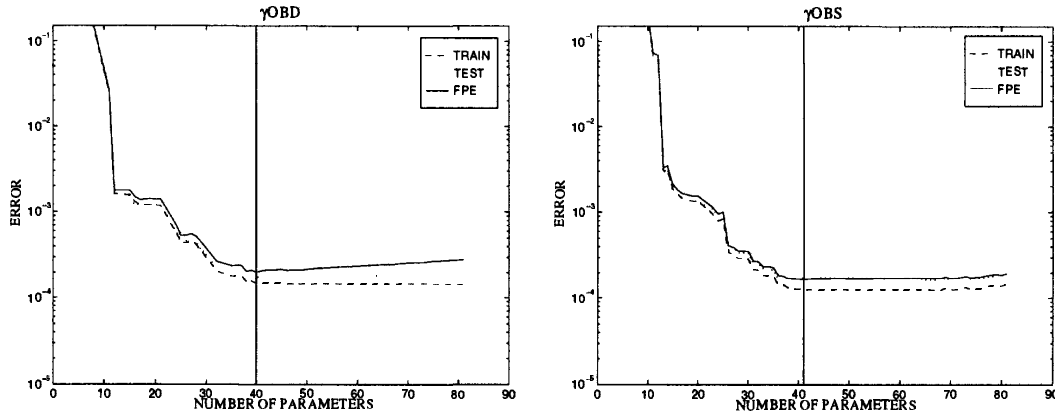

Figure 1: The evolution of training and test errors during pruning for the Mackey-Glass time series for a training set of size 250. In the left panel is shown pruning by γOBD, while in the right we show pruning by γOBS. The vertical solid line indicates the network for which the *estimated* test error is minimal.

test error, offset by a constant corresponding to the FPE-offset in the left panel of figure 1. The most important feature of this plot is that ranking according to the estimated test error is consistent with ranking according to the actual test error. In the right panel of figure 2, however, we see that γOBS highly underestimates the actual errors resulting from the quadratic retraining. It is not clear how the ranking inconsistencies affect the overall performance of γOBS. The weight selected for pruning (indicated by a circle) is clearly not the optimal according to the actual test error. However, as depicted in the figure, after full Gauss-Newton retraining for 20 epochs the measured actual test error is comparable to the estimated value (retraining is indicated by the arrow). Hence, one may say that γOBS "recovers" after retraining, while the initial estimate based on quadratic retraining is rather poor.

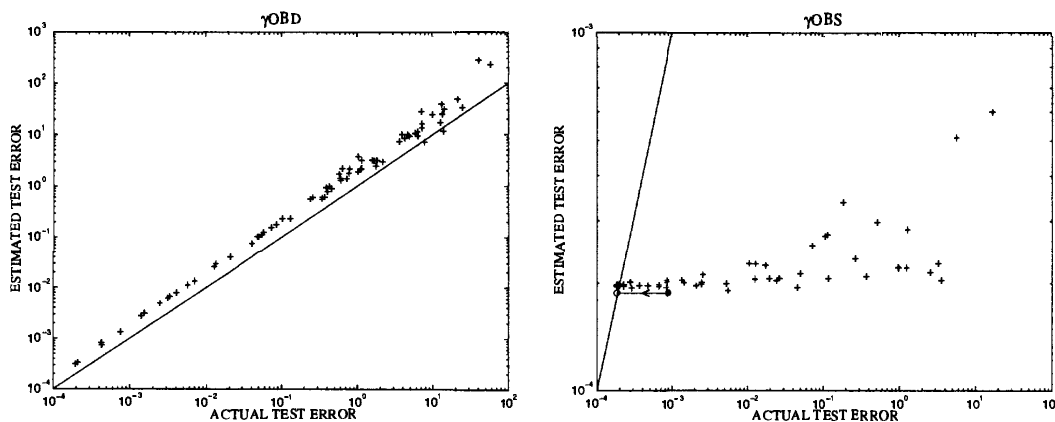

Figure 2: Left panel: Estimated test errors for fully connected network using γOBD and the actual test errors computed by actual deletion of the weight and computing the test error on the 8500 members test set. Right panel: Errors for fully connected network using γOBS. The weight selected for pruning is indicated by a circle, the result of further retraining is indicated by an arrow.

# 6   CONCLUSION

Since a main objective of pruning algorithms is to improve generalization we suggest that weight saliencies are estimated from the test error rather than the training error. We have shown how this might be carried out for scalar function approximation, in which case we have a rather simple test error estimate (based on Akaike's FPE). We provided implementation details for a scheme of linear complexity, $\gamma$OBD, which is the generalization of OBD and a scheme of quadratic complexity $\gamma$OBS which is the generalization of OBS. Furthermore, we provided a way to significantly reduce the computational overhead involved in the handling of nuisance parameters. An application within time series prediction showed the viability of the suggested approach.

## Acknowledgements

We thank Peter Magnus Nørgaard for valuable discussions. This research is supported by the Danish Natural Science and Technical Research Councils through the Computational Neural Network Center (CONNECT). JL acknowledge the Radioparts Foundation for financial support.

## Footnotes

[1] $\mathbf{R}$ will be a positive definite diagonal matrix.

[2]The expression is for the case of all weight decays being equal, see (Hansen and With, 94) for the general expression.

## References

H. Akaike: *Fitting Autoregressive Models for Prediction.* Ann. Inst. Stat. Mat. **21**, 243–247, (1969).

Y. Le Cun, J.S. Denker, and S.A. Solla: *Optimal Brain Damage.* In Advances in Neural Information Processing Systems 2, Morgan Kaufman, 598–605, (1990).

L.K. Hansen and M. With Petersen: *Controlled Growth of Cascade Correlation Nets.* Proceedings of ICANN'94 International Conference on Neural Networks, Sorrento, Italy, 1994. Eds. M. Marinaro and P.G. Morasso, 797–800, (1994).

B. Hassibi, D. G. Stork, and G. J. Wolff: *Optimal Brain Surgeon and General Network Pruning,* in Proceedings of the 1993 IEEE International Conference on Neural Networks, San Francisco (Eds. E.H. Ruspini et al. ) 293–299, (1993).

J. Larsen: *Design of Neural Network Filters.* Ph.D. Thesis, Electronics Institute, Technical University of Denmark, (1993).

J. Larsen and L.K. Hansen: *Generalization Performance of Regularized Neural Network Models.* "Neural Networks for Signal Processing IV" Proceedings of the IEEE Workshop, Eds. J. Vlontzos et al., IEEE Service Center, Piscataway NJ, 42–51, (1994).

J.E. Moody: *Note on Generalization, Regularization and Architecture Selection in Nonlinear Systems.* In Neural Networks For Signal Processing; Proceedings of the 1991 IEEE-SP Workshop, (Eds. B.H. Juang, S.Y. Kung, and C. Kamm), IEEE Service Center, 1–10, (1991).

C. Svarer, L.K. Hansen, and J. Larsen: *On Design and Evaluation of Tapped Delay Line Networks,* In Proceedings of the 1993 IEEE International Conference on Neural Networks, San Francisco, (Eds. E.H. Ruspini et al. ) 46–51, (1993).

A.S. Weigend, B.A. Huberman, and D.E. Rumelhart: *Prediction the future: A Connectionist Approach.* Int. J. of Neural Systems **3**, 193-209, (1990).

H. White: *Learning in Artificial Neural Networks: A Statistical Perspective.* Neural Computation **1**, 425-464, (1989).